# Shared Context Probabilistic Transducers

**Yoshua Bengio***
Dept. IRO,
Université de Montréal,
Montréal (QC), Canada, H3C 3J7
bengioy@iro.umontreal.ca

**Samy Bengio[†]**
Microcell Labs,
1250, René Lévesque Ouest,
Montréal (QC), Canada, H3B 4W8
samy.bengio@microcell.ca

**Jean-François Isabelle[‡]**
Microcell Labs,
1250, René Lévesque Ouest,
Montréal (QC), Canada, H3B 4W8
jean-francois.isabelle@microcell.ca

**Yoram Singer**
AT&T Laboratories,
Murray Hill, NJ 07733, USA,
singer@research.att.com

## Abstract

Recently, a model for supervised learning of probabilistic transducers represented by suffix trees was introduced. However, this algorithm tends to build very large trees, requiring very large amounts of computer memory. In this paper, we propose a new, more compact, transducer model in which one shares the parameters of distributions associated to contexts yielding similar conditional output distributions. We illustrate the advantages of the proposed algorithm with comparative experiments on inducing a noun phrase recognizer.

## 1 Introduction

Learning algorithms for sequential data modeling are important in many applications such as natural language processing and time-series analysis, in which one has to learn a model from one or more sequences of training data. Many of these algorithms can be cast as *weighted transducers* (Pereira, Riley and Sproat, 1994), which associate input sequences to output sequences, with weights for each input/output

sequence pair. When these weights are interpreted as probabilities, such models are called *probabilistic transducers*. In particular, a probabilistic transducer can represent the conditional probability distribution of output sequences given an input sequence. For example, algorithms for combining several transducers were found useful in natural language and speech processing (Riley and Pereira, 1994). Very often, weighted transducers use an intermediate variable that represents "context", such as the state variable of Hidden Markov Models (Baker, 1975; Jelinek, 1976). A particular type of weighted transducer, called *Input/Output Hidden Markov Model*, is one in which the input-to-context distribution and context-to-output distribution are represented by flexible parameterized models (such as neural networks) (Bengio and Frasconi, 1996). In this paper, we will study probabilistic transducers with a deterministic input-to-state mapping (i.e., a function from the past input subsequence to the current value of the context variable). One such transducer is the one which assigns a value of the context variable to every value of the past input subsequence already seen in the data. This input-to-state mapping can be efficiently represented by a tree. Such transducers are called *suffix tree transducers* (Singer, 1996).

A problem with suffix tree transducers is that they tend to yield very large trees (whose size may grow as $O(n^2)$ for a sequence of data of length $n$). For example, in the application studied in this paper, one obtains trees requiring over a gigabyte of memory. Heuristics may be used to limit the growth of the tree (e.g., by limiting the maximum depth of the context, i.e., of the tree, and by limiting the maximum number of contexts, i.e., nodes of the tree). In this paper, instead, we propose a new model for a probabilistic transducer with deterministic input-to-state function in which this function is compactly represented, by sharing parameters of contexts which are associated to similar output distributions. Another way to look at the proposed algorithm is that it searches for a clustering of the nodes of a suffix tree transducer. The data structure that represents the contexts is not anymore a tree but a single-root acyclic directed graph.

## 2  Background: Suffix Tree Probabilistic Transducers

The learning algorithm for suffix tree probabilistic transducers (Singer, 1996) constructs the model $P(y_1^n|x_1^n)$ from **discrete input** sequences $x_1^n = \{x_1, x_2, \ldots, x_n\}$ to output sequences $y_1^n = \{y_1, y_2, \ldots, y_n\}$, where $x_t$ are elements of a finite alphabet $\Sigma_{in}$. This distribution is represented by a tree in which each internal node may have a child for every element of $\Sigma_{in}$, therefore associating a label $\in \Sigma_{in}$ to each arc. A node at depth $d$ is labeled with the sequence $\sigma_1^d$ of labels on arcs from root to node, corresponding to a particular *input context*, e.g., at some position $n$ in the sequence a context of length $d$ is the value $\sigma_1^d$ of the preceding subsequence $x_{n-d+1}^n$. Each node at depth $d$ is therefore associated with a model of the output distribution in this context, $P(y_n|x_{n-d+1}^n = \sigma_1^d)$ (independent of $n$).

To obtain a local output probability for $y_n$ (i.e., given $x_1^n$), one follows the longest possible path from the root to a node a depth $d$ according to the labels $x_n$, $x_{n-1}$, ... $x_{n-d+1}$. The local output probability at this node is used to model $y_n$. Since $P(y_1^T|x_1^T)$ can always be written $\prod_{n=1}^T P(y_n|x_1^n)$, the overall input/output conditional distribution can be decomposed, according to this model, as follows:

$$P(y_1^T|x_1^T) = \prod_{n=1}^T P(y_n|x_{n-d(x_1^n)+1}^n), \tag{1}$$

where $d(x_1^n)$ is the depth of the node of the tree associated with the longest suffix $\sigma_1^d = x_{n-d+1}^n$ of $x_1^n$. Figure 1 gives a simple example of a suffix tree transducer.

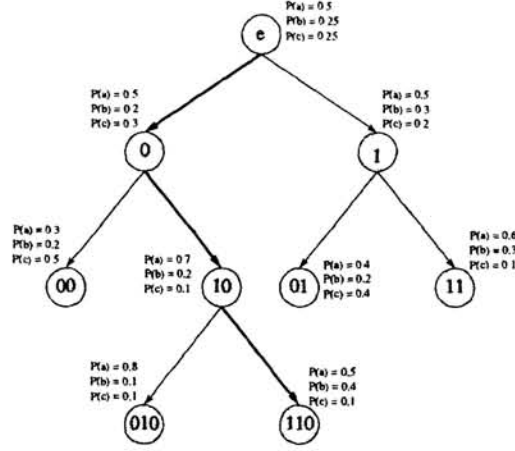

Figure 1: *Example of suffix tree transducer (Singer, 1996). The input alphabet,* $\Sigma_{in} = \{0, 1\}$ *and the output alphabet,* $\Sigma_{out} = \{a, b, c\}$. *For instance,* $P(a|00110) = P(a|110) = 0.5$.

## 3   Proposed Model and Learning Algorithm

In the model proposed here, the input/output conditional distribution $P(y_1^T|x_1^T)$ is represented by *a single-root acyclic directed graph*. Each *node* of this graph is associated with a *set of contexts* $C_{node} = \{\sigma_1^{d_i}\}$, corresponding to all the paths $i$ (of various lengths $d_i$) from the root of the tree to this *node*. All these contexts are associated with the same local output distribution $P(y_n|x_1^n$ has a suffix in $C_{node})$.

Like in suffix tree transducers, each internal node may have a child for every element of $\Sigma_{in}$. The arc is labeled with the corresponding element of $\Sigma_{in}$. Also like in suffix tree transducers, to obtain $P(y_n|x_1^n)$, one follows the path from the root to the deepest node called $deepest(x_1^n)$ according to the labels $x_n$, $x_{n-1}$, etc... The local output distribution at this node is used to predict $y_n$ or its probability. The overall conditional distribution is therefore given by

$$P(y_1^T|x_1^T) = \prod_{n=1}^{T} P(y_n|deepest(x_1^n)) \tag{2}$$

where the set of contexts $C_{deepest(x_1^n)}$ associated to the deepest node $deepest(x_1^n)$ contains a suffix of $x_1^n$. The model can be used both to compute the conditional probability of a given input/output sequence pair, or to guess an output sequence given an input sequence. Note that the input variable can contain delayed values of the output variable (as in Variable Length Markov Models).

### 3.1   Proposed Learning Algorithm

We present here a constructive learning algorithm for building the graph of the model and specify which data points are used to update each local output model (associated to nodes of the graph). The algorithm is on-line and operates according to two regimes: (1) adding new nodes and simply updating the local output distributions at existing nodes, and (2) merging parts of the graph which represent similar distributions. If there are multiple sequences in the training data they are concatenated in order to obtain a single input/output sequence pair.

(1) After every observation $(x_n, y_n)$, the algorithm updates the output distributions

of the nodes for which $C_{node(x_1^n)}$ contains a suffix of $x_1^n$, possibly adding new nodes (with labels $x_{n-d_*}^n$) until $x_1^n \in C_{node}$ for some *node*.

(2) Every $\tau_{\text{merge}}$ observations, the algorithm attempts to merge sub-graphs which are found similar enough, by comparing the $N(N-1)/2$ pairs of sub-graphs rooted at the $N$ nodes that have seen at least $\min_n$ observations. Merging two subgraphs is equivalent to forcing them to share parameters (as well as reducing the size of the representation of the distribution). A merge is performed between the graphs rooted at nodes $a$ and $b$ if $\Delta(a, b) < \min_\Delta$ and the merge succeeds. The details of the similarity measure and merging algorithm are given in the next subsections.

## 3.2   Similarity Measure Between Rooted Subgraphs

In order to compare (asymmetrically) output distributions $P(y|a)$ and $P(y|b)$ at two nodes $a$ and $b$, one can use the Kullback-Liebler divergence:

$$KL(a, b) = \sum_{y \in \Sigma_{out}} P(y|b) \log \frac{P(y|b)}{P(y|a)} \tag{3}$$

However, we want to compare the whole acyclic graphs rooted at these 2 nodes. In order to do so, let us define the following. Let $s$ be a string of input labels, and $b$ a node. Define $desc(b, s)$ as the most remote descendant of $b$ obtained by following from $b$ the arcs whose labels correspond to the sequence $s$. Let $descendents(a)$ be the set of strings obtained by following the arcs starting from node $a$ until reaching the leaves which have $a$ as an ancestor. Let $P(s|a)$ be the probability of following the arcs according to string $s$, starting from node $a$. This distribution can be estimated by counting the relative number of descendents through each of the children of each node.

To compare the graphs rooted at two nodes $a$ and $b$, we extend the KL divergence by weighing each of the descendents of $a$, as follows:

$$WKL(a, b) = \sum_{s \in descendents(a)} P(s|a) KL(desc(a, s), desc(b, s)) \tag{4}$$

Finally, to obtain a symmetric measure, we define

$$\Delta(a, b) = WKL(a, b) + WKL(b, a) \tag{5}$$

that is used in the merge phase of the constructive learning algorithm to decide whether the subgraphs rooted at $a$ and $b$ should be merged.

## 3.3   Merging Two Rooted Subgraphs

If $\Delta(a, b) < \min_\Delta$ (a predefined threshold) we want to merge the two subgraphs rooted at $a$ and $b$ and create a new subgraph rooted at $c$. The local output distribution at $c$ is obtained from the local output distributions at $a$ and $b$ as follows:

$$P(y_n|c) = P(y_n|a)P(a|a \text{ or } b) + P(y_n|b)P(b|a \text{ or } b) \tag{6}$$

where we define

$$P(a|a \text{ or } b) = \frac{\alpha^{d(a)}}{\alpha^{d(a)} + \alpha^{d(b)}}, \tag{7}$$

where $d(a)$ is the length of the longest path from the root to node $a$, and $\alpha$ represents a prior parameter (between 0 and 1) on the depth of the acyclic graphs. This prior parameter can be used to induce a prior distribution over possible rooted acyclic graphs structures which favors smaller graphs and shorter contexts (see the mixture of probabilistic transducers of (Singer, 1996)).

The merging algorithm can then be summarized as follows:

- The parents of $a$ and $b$ become parents for $c$.

- Some verifications are made to prevent merges which would yield to cycles in the graph. The nodes $a$ and $b$ are not merged if they are parents of one another.

- We make each child of $a$ a child of $c$. For each child $u$ of $b$ (following an arc labeled $l$), look for the corresponding child $v$ of $c$ (also following the arc labeled $l$). If there is no such child, and $u$ is not a parent of $c$, make $u$ a new child of $c$. Else, if $u$ and $v$ are not parents of each other, recursively merge them.

- Delete nodes $a$ and $b$, as well as all the links from and to these nodes.

This algorithm is symmetric with respect to $a$ and $b$ except when a merge cannot be done because $a$ and $b$ are parents of one another. In this case, an asymmetric decision must be taken: we chose to keep only $a$ and reject $b$. Figure 2 gives a simple example of merge.

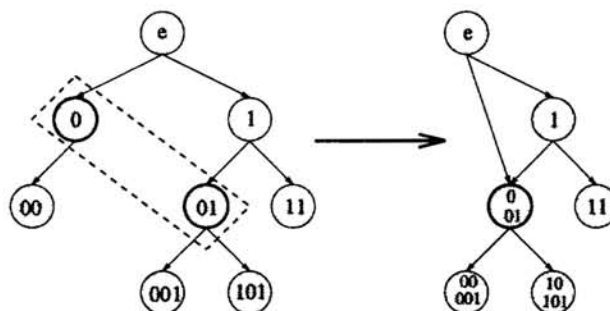

Figure 2: *This figure shows how two nodes are merged. The result is no longer a tree, but a directed graph. Some verifications are done to avoid cycles in the graph. Each node can have multiple labels, corresponding to the multiple possible paths from the root to the node.*

# 4   Comparative Experiments

We compared experimentally our model to the one proposed in (Singer, 1996) on mixtures of suffix tree transducers, using the same task. Given a text where each word is assigned an appropriate part-of-speech value (verb, noun, adjective, etc), the task is to identify the noun phrases in the text. The UPENN tree-bank corpus database was used in these experiments. The input vocabulary size, $|\Sigma_{in}| = 41$, is the number of possible part-of-speech tags, and the output vocabulary size is $|\Sigma_{out}| = 2$. The model was trained over 250000 marked tags, constraining the tree to be of maximal depth 15. The model was then tested (freezing the model structure and its parameters) over 37000 other tags. Using the mixture of suffix tree transducers (Singer, 1996) and thresholding the output probability at 0.5 to take output decisions, yielded an accuracy rate of 97.6% on the test set, but required over 1 gigabyte of computer memory.

To make interesting comparisons with the shared context transducers, we chose the following experimental scheme. Not only did we fix the maximal depth of the directed graph to 15, but we also fixed the maximal number of allocated nodes, i.e., simulating fixed memory resources. When this number was reached, we froze the structure but continued to update the parameters of the model until the end of the training database was reached. For the shared context version, whenever a merge freed some nodes, we let the graph grow again to its maximal node size. At the end of this process, we evaluated the model on the test set.

We tested this method for various values of the maximum number of nodes in the graph. For each experiment, we tried different values of the other parameters (the similarity threshold $min_\Delta$ for merging, the minimum number of observations $min_n$ at a node before it can be considered for a merge, and the delay $\tau_{merge}$ between two merging phases), and we picked the one which performed the best *on the training set*. Results are reported in figure 3.

| maximal number of nodes | with merge (%) | without merge (%) |
|---|---|---|
| 20 | 0.762 | 0.584 |
| 50 | 0.827 | 0.624 |
| 100 | 0.861 | 0.727 |
| 500 | 0.924 | 0.867 |
| 1000 | 0.949 | 0.917 |
| 2000 | 0.949 | 0.935 |
| 5000 | 0.952 | 0.948 |

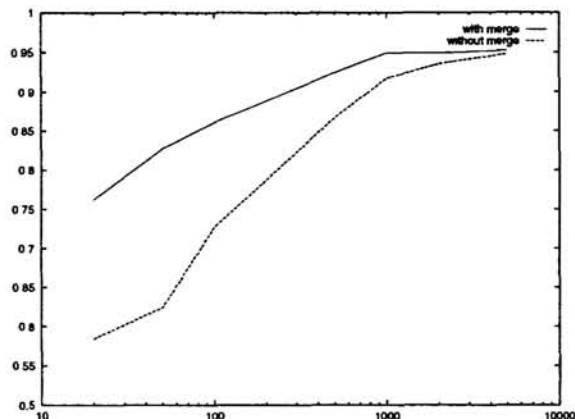

Figure 3: *This figure shows the generalization accuracy rate of a transducer with merges (shared contexts graph) against one without merges (suffix tree), with different maximum number of nodes. The maximum number of nodes are in a logarithmic scale, and the accuracy rates are expressed in relative frequency of correct classification.*

As can be seen from the results, the accuracy rate over the test set is better for transducers with shared contexts than without. More precisely, the gain is greater when the maximum number of nodes is smaller. When we fix the maximum number of nodes to a very small value (20), a shared context transducer performs 1.3 times better (in classification error) than a non-shared one. This gain becomes smaller and smaller as the maximum size increases. Beyond a certain maximum size, there is almost no gain, and one could probably observe a loss for some large sizes. We also need to keep in mind that the larger the transducer is, the slower the program to create the shared context transducer is, compared to the non-shared one. Finally, it is interesting to note that using only 5000 nodes, we were able to obtain 95.2% accuracy, which is only 2.4% less than those obtained with no constraint on the number of nodes.

## 5   Conclusion

In this paper, we have presented the following:

- A new probabilistic model for probabilistic transducers with deterministic input-to-state function, represented by a rooted acyclic directed graph with nodes associated to a set of contexts and children associated to the different input symbols. This is a generalization of the suffix tree transducer.

- A constructive learning algorithm for this model, based on construction and merging phases. The merging is obtained by clustering parts of the graph which represent a similar conditional distribution.

- Experimental results on a natural-language task showing that when the size of the graph is constrained, this algorithm performs better than the purely constructive (no merge) suffix tree algorithm.

## Footnotes

* Yoshua Bengio is also with AT&T Laboratories, Holmdel, NJ 07733, USA.

† This work was performed while Samy Bengio was at INRS-Télécommunication, Ile-des-Soeurs, Québec, Canada, H3E 1H6

‡ This work was performed while Jean-François Isabelle was at INRS-Télécommunication, Ile-des-Soeurs, Québec, Canada, H3E 1H6

# References

Baker, J. (1975). Stochastic modeling for automatic speech understanding. In Reddy, D., editor, *Speech Recognition*, pages 521–542. Academic Press, New York.

Bengio, S. and Bengio, Y. (1996). An EM algorithm for asynchronous input/output hidden markov models. In *Proceedings of the International Conference on Neural Information Processing*, Honk Kong.

Bengio, Y. and Frasconi, P. (1996). Input/Output HMMs for sequence processing. *IEEE Transactions on Neural Networks*, 7(5):1231–1249.

Jelinek, F. (1976). Continuous speech recognition by statistical methods. *Proceedings of the IEEE*, 64:532–556.

Pereira, F., Riley, M., and Sproat, R. (1994). Weighted rational transductions and their application to human language processing. In *ARPA Natural Language Processing Workshop*.

Riley, M. and Pereira, F. (1994). Weighted-finite-automata tools with applications to speech and language processing. Technical Report Technical Memorandum 11222-931130-28TM, AT&T Bell Laboratories.

Singer, Y. (1996). Adaptive mixtures of probabilistic transducers. In Mozer, M., Touretzky, D., and Perrone, M., editors, *Advances in Neural Information Processing Systems 8*. MIT Press, Cambridge, MA.
